# Generalization to Unseen Cases

**Teemu Roos**
Helsinki Institute for Information Technology
P.O.Box 68, 00014 Univ. of Helsinki, Finland
teemu.roos@cs.helsinki.fi

**Peter Grünwald**
CWI, P.O.Box 94079, 1090 GB,
Amsterdam, The Netherlands
pdg@cwi.nl

**Petri Myllymäki**
Helsinki Institute for Information Technology
P.O.Box 68, 00014 Univ of Helsinki, Finland
petri.myllymaki@cs.helsinki.fi

**Henry Tirri**
Nokia Research Center
P.O.Box 407 Nokia Group, Finland
henry.tirri@nokia.com

## Abstract

We analyze classification error on unseen cases, i.e. cases that are different from those in the training set. Unlike standard generalization error, this *off-training-set error* may differ significantly from the empirical error with high probability even with large sample sizes. We derive a data-dependent bound on the difference between off-training-set and standard generalization error. Our result is based on a new bound on the missing mass, which for small samples is stronger than existing bounds based on Good-Turing estimators. As we demonstrate on UCI data-sets, our bound gives nontrivial generalization guarantees in many practical cases. In light of these results, we show that certain claims made in the No Free Lunch literature are overly pessimistic.

## 1 Introduction

A large part of learning theory deals with methods that bound the generalization error of hypotheses in terms of their empirical errors. The standard definition of generalization error allows overlap between the training sample and test cases. When such overlap is not allowed, i.e., when considering *off-training-set error* [1]–[5] defined in terms of only previously unseen cases, usual generalization bounds do not apply. The off-training-set error and the empirical error sometimes differ significantly with high probability even for large sample sizes. In this paper, we show that in many practical cases, one can nevertheless bound this difference. In particular, we show that with high probability, in the realistic situation where the number of *repeated* cases, or duplicates, relative to the total sample size is small, the difference between the off-training-set error and the standard generalization error is also small. In this case *any* standard generalization error bound, no matter how it is arrived at, transforms into a similar bound on the off-training-set error.

**Our Contribution**  We show that with probability at least $1-\delta$, if there are $r$ repetitions in the training sample, then the difference between the off-training-set error and the standard generalization error is at most of order $O\left(\sqrt{\frac{1}{n}\left(\log\frac{4}{\delta} + r\log n\right)}\right)$ (Thm. 2). Our main

result (Corollary 1 of Thm. 1) gives a stronger non-asymptotic bound that can be evaluated numerically. The proof of Thms. 1 and 2 is based on Lemma 2, which is of independent interest, giving a new lower bound on the so-called *missing mass*, the total probability of as yet unseen cases. For small samples and few repetitions, this bound is significantly stronger than existing bounds based on Good-Turing estimators [6]–[8].

**Properties of Our Bounds** Our bounds hold (1) *uniformly*, are (2) *distribution-free* and (3) *data-dependent*, yet (4) *relevant for data-sets encountered in practice*. Let us consider these properties in turn. Our bounds hold uniformly in that they hold for *all* hypotheses (functions from features to labels) at the same time. Thus, unlike many bounds on standard generalization error, our bounds do not depend in any way on the richness of the hypothesis class under consideration measured in terms of, for instance, its VC dimension, or the margin of the selected hypothesis on the training sample, or any other property of the mechanism with which the hypothesis is chosen. Our bounds are distribution-free in that they hold no matter what the (unknown) data-generating distribution is. Our bounds depend on the *data*: they are useful only if the number of repetitions in the training set is very small compared to the training set size. However, in machine learning practice this is often the case as demonstrated in Sec. 3 with several UCI data-sets.

**Relevance** Why are our results interesting? There are at least three reasons, the first two of which we discuss extensively in Sec. 4: (1) The use of off-training-set error is an essential ingredient of the No Free Lunch (NFL) theorems [1]–[5]. Our results counter-balance some of the overly pessimistic conclusions of this work. This is all the more relevant since the NFL theorems have been quite influential in shaping the thinking of both theoretical and practical machine learning researchers (see, e.g., Sec. 9.2 of the well-known textbook [5]). (2) The off-training-set error is an intuitive measure of generalization performance. Yet in practice it differs from standard generalization error (even with continuous feature spaces). Thus, we feel, it is worth studying. (3) Technically, we establish a surprising connection between off-training-set error (a concept from classification) and missing mass (a concept mostly applied in language modeling), and give a new lower bound on the missing mass.

The paper is organized as follows: In Sec. 2 we fix notation, including the various error functionals considered, and state some preliminary results. In Sec. 3 we state our bounds, and we demonstrate their use on data-sets from the UCI machine learning repository. We discuss the implications of our results in Sec. 4. Postponed proofs are in Appendix A.

## 2   Preliminaries and Notation

Let $\mathcal{X}$ be an arbitrary space of inputs, and let $\mathcal{Y}$ be a discrete space of labels. A learner observes a random *training sample*, $D$, of size $n$, consisting of the values of a sequence of input–label pairs $((X_1, Y_1), ..., (X_n, Y_n))$, where $(X_i, Y_i) \in \mathcal{X} \times \mathcal{Y}$. Based on the sample, the learner outputs a hypothesis $h : \mathcal{X} \to \mathcal{Y}$ that gives, for each possible input value, a prediction of the corresponding label. The learner is successful if the produced hypothesis has high probability of making a correct prediction when applied to a test case. $(X_{n+1}, Y_{n+1})$. Both the training sample and the test case are independently drawn from a common *generating distribution* $P^*$. We use the following error functionals:

**Definition 1 (errors).** *Given a training sample $D$ of size $n$, the* i.i.d., off-training-set, *and* empirical error *of a hypothesis $h$ are given by*

$$
\begin{aligned}
\mathcal{E}_{\text{iid}}(h) &:= \Pr[Y \neq h(X)] & \textit{i.i.d. error}, \\
\mathcal{E}_{\text{ots}}(h, D) &:= \Pr[Y \neq h(X) \mid X \notin \mathcal{X}_D] & \textit{off-training-set error}, \\
\mathcal{E}_{\text{emp}}(h, D) &:= \tfrac{1}{n} \sum_{i=1}^{n} \mathbb{I}_{\{h(X_i) \neq Y_i\}} & \textit{empirical error},
\end{aligned}
$$

*where $\mathcal{X}_D$ is the set of $X$-values occurring in sample $D$, and the indicator function $\mathbb{I}_{\{\cdot\}}$ takes value one if its argument is true and zero otherwise.*

The first one of these is just the standard generalization error of learning theory. Following [2], we call it i.i.d. error. For general input spaces and generating distributions $\mathcal{E}_{\mathrm{ots}}(h, D)$ may be undefined for some $D$. In either case, this is not a problem. First, if $\mathcal{X}_D$ has measure one, the off-training-set error is undefined and we need not concern ourselves with it; the relevant error measure is $\mathcal{E}_{\mathrm{iid}}(h)$ and standard results apply[1]. If, on the other hand, $\mathcal{X}_D$ has measure zero, the off-training-set error and the i.i.d. error are equivalent and our results (in Sec. 3 below) hold trivially. Thus, *if* off-training-set error is relevant, our results hold.

**Definition 2.** *Given a training sample $D$, the* sample coverage $p(\mathcal{X}_D)$ *is the probability that a new $X$-value appears in $D$: $p(\mathcal{X}_D) := \Pr[X \in \mathcal{X}_D]$, where $\mathcal{X}_D$ is as in Def. 1. The remaining probability, $1 - p(\mathcal{X}_D)$, is called the* missing mass.

**Lemma 1.** *For any training set $D$ such that $\mathcal{E}_{\mathrm{ots}}(h, D)$ is defined, we have*

$$a) \quad |\mathcal{E}_{\mathrm{ots}}(h, D) - \mathcal{E}_{\mathrm{iid}}(h)| \leq p(\mathcal{X}_D) \ ,$$
$$b) \quad \mathcal{E}_{\mathrm{ots}}(h, D) - \mathcal{E}_{\mathrm{iid}}(h) \leq \frac{p(\mathcal{X}_D)}{1 - p(\mathcal{X}_D)} \mathcal{E}_{\mathrm{iid}}(h) \ .$$

*Proof.* Both bounds follow essentially from the following inequalities[2]:

$$
\begin{aligned}
\mathcal{E}_{\mathrm{ots}}(h, D) \quad &= \frac{\Pr[Y \neq h(X), X \notin \mathcal{X}_D]}{\Pr[X \notin \mathcal{X}_D]} \leq \frac{\Pr[Y \neq h(X)]}{\Pr[X \notin \mathcal{X}_D]} \wedge 1 = \frac{\mathcal{E}_{\mathrm{iid}}(h)}{1 - p(\mathcal{X}_D)} \wedge 1 \\
&= \left( \frac{\mathcal{E}_{\mathrm{iid}}(h)}{1 - p(\mathcal{X}_D)} \wedge 1 \right) (1 - p(\mathcal{X}_D)) + \left( \frac{\mathcal{E}_{\mathrm{iid}}(h)}{1 - p(\mathcal{X}_D)} \wedge 1 \right) p(\mathcal{X}_D) \\
&\leq \mathcal{E}_{\mathrm{iid}}(h) + p(\mathcal{X}_D) \ ,
\end{aligned}
$$

where $\wedge$ denotes the minimum. This gives one direction of Lemma 1.a (an *upper* bound on $\mathcal{E}_{\mathrm{ots}}(h, D)$); the other direction is obtained by using analogous inequalities for the quantity $1 - \mathcal{E}_{\mathrm{ots}}(h, D)$, with $Y \neq h(X)$ replaced by $Y = h(X)$, which gives the upper bound $1 - \mathcal{E}_{\mathrm{ots}}(h, D) \leq 1 - \mathcal{E}_{\mathrm{iid}}(h) + p(\mathcal{X}_D)$. Lemma 1.b follows from the first line by ignoring the upper bound 1, and subtracting $\mathcal{E}_{\mathrm{iid}}(h)$ from both sides. □

Given the value of (or an upper bound on) $\mathcal{E}_{\mathrm{iid}}(h)$, the upper bound of Lemma 1.b may be significantly stronger than that of Lemma 1.a. However, in this work we only use Lemma 1.a for simplicity since it depends on $p(\mathcal{X}_D)$ alone. The lemma would be of little use without a good enough upper bound on the sample coverage $p(\mathcal{X}_D)$, or equivalently, a lower bound on the missing mass. In the next section we obtain such a bound.

## 3 An Off-training-set Error Bound

Good-Turing estimators [6], named after Irving J. Good, and Alan Turing, are widely used in language modeling to estimate the missing mass. The known small bias of such estima-tors, together with a rate of convergence, can be used to obtain lower and upper bound for the missing mass [7, 8]. Unfortunately, for the sample sizes we are interested in, the lower bounds are not quite tight enough (see Fig. 1 below). In this section we state a new lower bound, not based on Good-Turing estimators, that is practically useful in our context. We compare this bound to the existing ones after Thm. 2.

Let $\bar{\mathcal{X}}_n \subset \mathcal{X}$ be the set consisting of the $n$ most probable individual values of $X$. In case there are several such subsets any one of them will do. In case $\mathcal{X}$ has less than $n$ elements, $\bar{\mathcal{X}}_n := \mathcal{X}$. Denote for short $\bar{p}_n := \Pr[X \in \bar{\mathcal{X}}_n]$. No assumptions are made regarding the value of $\bar{p}_n$, it may or may not be zero. The reason for us being interested in $\bar{p}_n$ is that

it gives us an upper bound $p(\mathcal{X}_D) \leq \bar{p}_n$ on the sample coverage that holds for all $D$. We prove that when $\bar{p}_n$ is large it is likely that a sample of size $n$ will have several repeated $X$-values so that the number of distinct $X$-values is less than $n$. This implies that if a sample with a small number of repeated $X$-values is observed, it is safe to assume that $\bar{p}_n$ is small and therefore, the sample coverage $p(\mathcal{X}_D)$ must also be small.

**Lemma 2.** *The probability of obtaining a sample of size $n \geq 1$ with at most $0 \leq r < n$ repeated $X$-values is upper-bounded by* $\Pr[\text{``at most } r \text{ repetitions''}] \leq \Delta(n, r, \bar{p}_n)$ *, where*

$$\Delta(n, r, \bar{p}_n) := \sum_{k=0}^{n} \binom{n}{k} \bar{p}_n^k (1 - \bar{p}_n)^{n-k} f(n, r, k) \tag{1}$$

*and $f(n, r, k)$ is given by* $f(n, r, k) := \begin{cases} 1 & \text{if } k < r \\ \min\left(\binom{k}{r} \frac{n!}{(n-k+r)!} n^{-(k-r)}, 1\right) & \text{if } k \geq r. \end{cases}$

$\Delta(n, r, \bar{p}_n)$ *is a non-increasing function of $\bar{p}_n$.*

For a proof, see Appendix A. Given a fixed confidence level $1 - \delta$ we can now define a data-dependent upper bound on the sample coverage

$$\mathcal{B}(\delta, D) := \arg\min_p \{p \; : \; \Delta(n, r, p) \leq \delta\} \; , \tag{2}$$

where $r$ is the number of repeated $X$-values in $D$, and $\Delta(n, r, p)$ is given by Eq. (1).

**Theorem 1.** *For any $0 \leq \delta \leq 1$, the upper bound $\mathcal{B}(\delta, D)$ on the sample coverage given by Eq. (2) holds with at least probability $1 - \delta$:*

$$\Pr\left[p(\mathcal{X}_D) \leq \mathcal{B}(\delta, D)\right] \geq 1 - \delta \; .$$

*Proof.* Consider fixed values of the confidence level $1 - \delta$, sample size $n$, and probability $\bar{p}_n$. Let $R$ be the largest integer for which $\Delta(n, R, \bar{p}_n) \leq \delta$. By Lemma 2 the probability of obtaining at most $R$ repetitions is upper-bounded by $\delta$. Thus, it is sufficient that the bound holds whenever the number of repetitions is greater than $R$. For any such $r > R$, we have $\Delta(n, r, \bar{p}_n) > \delta$. By Lemma 2 the function $\Delta(n, r, \bar{p}_n)$ is non-increasing in $\bar{p}_n$, and hence it must be that $\bar{p}_n < \arg\min_p\{p \; : \; \Delta(n, r, p) \leq \delta\} = \mathcal{B}(\delta, D)$. Since $p(\mathcal{X}_D) \leq \bar{p}_n$, the bound then holds for all $r > R$. $\qquad\square$

Rather than the sample coverage $p(\mathcal{X}_D)$, the real interest is often in off-training-set error. Using the relation between the two quantities, one gets the following corollary that follows directly from Lemma 1.a and Thm. 1.

**Corollary 1 (main result: off-training-set error bound).** *For any $0 \leq \delta \leq 1$, the difference between the i.i.d. error and the off-training-set error is bounded by*

$$\Pr\left[\forall h \; |\mathcal{E}_{\text{ots}}(h, D) - \mathcal{E}_{\text{iid}}(h)| \leq \mathcal{B}(\delta, D)\right] \geq 1 - \delta \; .$$

Corollary 1 implies that the off-training-set error and the i.i.d. error are entangled, thus transforming all distribution-free bounds on the i.i.d. error to similar bounds on the off-training-set error. Since the probabilistic part of the result (Lemma 1) does not involve a specific hypothesis, Corollary 1 holds for all hypotheses at the same time, and does not depend on the richness of the hypothesis class in terms of, for instance, its VC dimension.

Figure 1 illustrates the behavior of the bound (2) as the sample size grows. It can be seen that for a small number of repetitions the bound is nontrivial already at moderate sample sizes. Moreover, the effect of repetitions is tolerable, and it diminishes as the number of repetitions grows. Table 1 lists values of the bound for a number of data-sets from the UCI machine learning repository [9]. In many cases the bound is about 0.10–0.20 or less.

Theorem 2 gives an upper bound on the rate with which the bound decreases as $n$ grows.

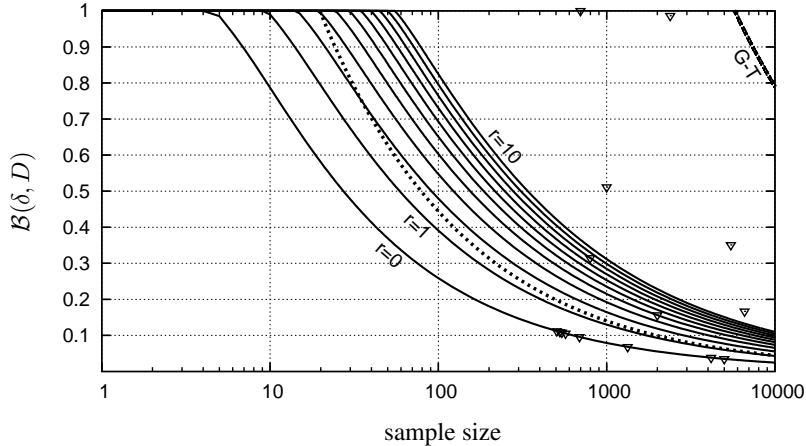

Figure 1: Upper bound $\mathcal{B}(\delta, D)$ given by Eq. (2) for samples with zero ($r = 0$) to ten ($r = 10$) repeated $X$-values on the 95 % confidence level ($\delta = 0.05$). The dotted curve is an asymptotic version for $r = 0$ given by Thm. 2. The curve labeled 'G-T' (for $r = 0$) is based on Good-Turing estimators (Thm. 3 in [7]). Asymptotically, it exceeds our $r = 0$ bound by a factor $O(\log n)$. Bound for the UCI data-sets in Table 1 are marked with small triangles ($\triangledown$). Note the log-scale for sample size.

**Theorem 2 (a weaker bound in closed-form).** *For all $n$ and all $\bar{p}_n$, all $r < n$, the function $\mathcal{B}(\delta, D)$ has the upper bound $\mathcal{B}(\delta, D) \leq 3\sqrt{\frac{1}{2n}\left(\log \frac{4}{\delta} + 2r \log n\right)}$.*

For a proof, see Appendix A. Let us compare Thm. 2 to the existing bounds on $\mathcal{B}(\delta, D)$ based on Good-Turing estimators [7, 8]. For fixed $\delta$, Thm. 3 in [7] gives an upper bound of $O\left(r/n + \log n/\sqrt{n}\right)$. The exact bound is drawn as the G-T curve in Fig. 1. In contrast, our bound gives $O\left(\sqrt{C + r \log n}/\sqrt{n}\right)$, for a known constant $C > 0$. For fixed $r$ and increasing $n$, this gives an improvement over the G-T bound of order $O(\log n)$ if $r = 0$, and $O(\sqrt{\log n})$ if $r > 0$. For $r$ growing faster than $O(\sqrt{\log n})$, asymptotically our bound becomes uncompetitive[3]. The real advantage of our bound is that, in contrast to G-T, it gives nontrivial bounds for sample sizes and number of repetitions that typically occur in classification problems. For practical applications in language modeling (large samples, many repetitions), the existing G-T bound of [7] is probably preferable.

The developments in [8] are also relevant, albeit in a more indirect manner. In Thm. 10 of that paper, it is shown that the probability that the missing mass is larger than its expected value by an amount $\epsilon$ is bounded by $e^{-(e/2)n\epsilon^2}$. In [7], Sec. 4, some techniques are developed to bound the expected missing mass in terms of the number of repetitions in the sample. One might conjecture that, combined with Thm. 10 of [8], these techniques can be extended to yield an upper bound on $\mathcal{B}(\delta, D)$ of order $O(r/n + 1/\sqrt{n})$ that would be asymptotically stronger than the current bound. We plan to investigate this and other potential ways to improve the bounds in future work. Any advance in this direction makes the implications of our bounds even more compelling.

Table 1: Bounds on the difference between the i.i.d. error and the off-training-set error given by Eq. (2) on confidence level 95% ($\delta = 0.05$). A dash (-) indicates no repetitions. Bounds greater than 0.5 are in parentheses.

| DATA | SAMPLE SIZE | REPETITIONS | BOUND |
|---|---|---|---|
| Abalone | 4177 | - | 0.0383 |
| Adult | 32562 | 25 | 0.0959 |
| Annealing | 798 | 8 | 0.3149 |
| Artificial Characters | 1000 | 34 | (0.5112) |
| Breast Cancer (Diagnostic) | 569 | - | 0.1057 |
| Breast Cancer (Original) | 699 | 236 | (1.0) |
| Credit Approval | 690 | - | 0.0958 |
| Cylinder Bands | 542 | - | 0.1084 |
| Housing | 506 | - | 0.1123 |
| Internet Advertisement | 2385 | 441 | (0.9865) |
| Isolated Letter Speech Recogn. | 1332 | - | 0.0685 |
| Letter Recognition | 20000 | 1332 | (0.6503) |
| Multiple Features | 2000 | 4 | 0.1563 |
| Musk | 6598 | 17 | 0.1671 |
| Page Blocks | 5473 | 80 | 0.3509 |
| Water Treatment Plant | 527 | - | 0.1099 |
| Waveform | 5000 | - | 0.0350 |

## 4  Discussion – Implications of Our Results

The use of off-training-set error is an essential ingredient of the influential No Free Lunch theorems [1]–[5]. Our results imply that, while the NFL theorems themselves are valid, some of the conclusions drawn from them are overly pessimistic, and should be reconsidered. For instance, it has been suggested that the tools of conventional learning theory (dealing with standard generalization error) are "ill-suited for investigating off-training-set error" [3]. With the help of the little add-on we provide in this paper (Corollary 1), *any* bound on standard generalization error can be converted to a bound on off-training-set error. Our empirical results on UCI data-sets show that the resulting bound is often not essentially weaker than the original one. Thus, the conventional tools turn out not to be so 'ill-suited' after all. Secondly, contrary to what is sometimes suggested[4], we show that one *can* relate performance on the training sample to performance on as yet unseen cases.

On the other side of the debate, it has sometimes been claimed that the off-training-set error is irrelevant to much of modern learning theory where often the feature space is continuous. This may seem to imply that off-training-set error coincides with standard generalization error (see remark after Def. 1). However, this is true only if the associated *distribution* is continuous: *then* the probability of observing the same $X$-value twice is zero. However, in practice even when the feature space has continuous components, data-sets sometimes contain repetitions (e.g., Adult, see Table 1), if only for the reason that continuous features may be discretized or truncated. In practice repetitions occur in many data-sets, implying that off-training-set error can be different from the standard i.i.d. error. Thus, off-training-set error is *relevant*. Also, it measures a quantity that is in some ways close to the meaning of 'inductive generalization' – in dictionaries the words 'induction' and 'generalization' frequently refer to 'unseen instances'. Thus, off-training-set error is not just relevant but also *intuitive*. This makes it all the more interesting that standard generalization bounds transfer to off-training-set error – and that is the central implication of this paper.

## Acknowledgments

We thank Gilles Blanchard for useful discussions. Part of this work was carried out while the first author was visiting CWI. This work was supported in part by the Academy of Finland (Minos, Prima), Nuffic, and IST Programme of the European Community, under the PASCAL Network, IST-2002-506778. This publication only reflects the authors' views.

## Footnotes

[1]Note however, that a continuous feature space does not necessarily imply this, see Sec. 4.

[2]This neat proof is due to Gilles Blanchard (personal communication).

[3]If data are i.i.d. according to a fixed $P^*$, then, as follows from the strong law of large numbers, $r$, considered as a function of $n$, will either remain zero for ever or will be larger than $cn$ for some $c > 0$, for all $n$ larger than some $n_0$. In practice, our bound is still relevant because typical data-sets often have $r$ very small compared to $n$ (see Table 1). This is possible because apparently $n \ll n_0$.

[4]For instance, "if we are interested in the error for [unseen cases], the NFL theorems tell us that (in the absence of prior assumptions) [empirical error] is meaningless" [2].

## References

[1] Wolpert, D.H.: On the connection between in-sample testing and generalization error. Complex Systems **6** (1992) 47–94

[2] Wolpert, D.H.: The lack of *a priori* distinctions between learning algorithms. Neural Computation **8** (1996) 1341–1390

[3] Wolpert, D.H.: The supervised learning no-free-lunch theorems. In: Proc. 6th Online World Conf. on Soft Computing in Industrial Applications (2001).

[4] Schaffer, C.: A conservation law for generalization performance. In: Proc. 11th Int. Conf. on Machine Learning (1994) 259–265

[5] Duda, R.O., Hart, P.E., Stork, D.G.: *Pattern Classification*, 2nd Edition. Wiley, 2001.

[6] Good, I.J.: The population frequencies of species and the estimation of population parameters. Biometrika **40** (1953) 237–264

[7] McAllester, D.A., Schapire, R.E.: On the convergence rate of Good-Turing estimators. In: Proc. 13th Ann. Conf. on Computational Learning Theory (2000) 1–6

[8] McAllester, D.A., Ortiz L.: Concentration inequalities for the missing mass and for histogram rule error. Journal of Machine Learning Research **4** (2003) 895–911.

[9] Blake, C., and Merz, C.: UCI repository of machine learning databases. Univ. of California, Dept. of Information and Computer Science (1998)

## A  Postponed Proofs

We first state two propositions that are useful in the proof of Lemma 2.

**Proposition 1.** *Let $\mathcal{X}_m$ be a domain of size $m$, and let $P^*_{\mathcal{X}_m}$ be an associated probability distribution. The probability of getting no repetitions when sampling $1 \leq k \leq m$ items with replacement from distribution $P^*_{\mathcal{X}_m}$ is upper-bounded by*

$$\Pr[\text{``no repetitions''} \mid k] \leq \frac{m!}{(m-k)!m^k} \ .$$

*Proof Sketch of Proposition 1.* By way of contradiction it is possible to show that the probability of obtaining no repetitions is maximized when $P^*_{\mathcal{X}_m}$ is uniform. After this, it is easily seen that the maximal probability equals the right-hand side of the inequality. □

**Proposition 2.** *Let $\mathcal{X}_m$ be a domain of size $m$, and let $P^*_{\mathcal{X}_m}$ be an associated probability distribution. The probability of getting at most $r \geq 0$ repeated values when sampling $1 \leq k \leq m$ items with replacement from distribution $P^*_{\mathcal{X}_m}$ is upper-bounded by*

$$\Pr[\text{``at most } r \text{ repetitions''} \mid k] \leq \begin{cases} 1 & \text{if } k < r \\ \min\left(\binom{k}{r}\frac{m!}{(m-k+r)!}m^{-(k-r)}, 1\right) & \text{if } k \geq r. \end{cases}$$

*Proof of Proposition 2.* The case $k < r$ is trivial. For $k \geq r$, the event "at most $r$ repetitions in $k$ draws" is equivalent to the event that there is at least one subset of size $k - r$ of the $X$-variables $\{X_1, \ldots, X_k\}$ such that all variables in the subset take distinct values. For a subset of size $k - r$, Proposition 1 implies that the probability that all values are distinct is at most $\frac{m!}{(m-k+r)!}m^{-(k-r)}$. Since there are $\binom{k}{r}$ subsets of the $X$-variables of size $k - r$, the union bound implies that multiplying this by $\binom{k}{r}$ gives the required result. □

*Proof of Lemma 2.* The probability of getting at most $r$ repeated $X$-values can be upper bounded by considering repetitions in the maximally probable set $\bar{\mathcal{X}}_n$ only. The probability of no repetitions in $\bar{\mathcal{X}}_n$ can be broken into $n+1$ mutually exclusive cases depending on how many $X$-values fall into the set $\bar{\mathcal{X}}_n$. Thus we get

$$\Pr[\text{``at most } r \text{ repetitions in } \bar{\mathcal{X}}_n\text{''}] = \sum_{k=0}^{n} \Pr[\text{``at most } r \text{ repetitions in } \bar{\mathcal{X}}_n\text{''} \mid k]\Pr[k] \ ,$$

where $\Pr[\cdot \mid k]$ denotes probability under the condition that $k$ of the $n$ cases fall into $\bar{\mathcal{X}}_n$, and $\Pr[k]$ denotes the probability of the latter occurring. Proposition 2 gives an upper bound on the conditional probability. The probability $\Pr[k]$ is given by the binomial distribution with parameter $\bar{p}_n$: $\Pr[k] = \text{Bin}(k \ ; \ n, \bar{p}_n) = \binom{n}{k}\bar{p}_n^k(1-\bar{p}_n)^{n-k}$ . Combining these gives the formula for $\Delta(n, r, \bar{p}_n)$. Showing that $\Delta(n, r, \bar{p}_n)$ is non-increasing in $\bar{p}_n$ is tedious but uninteresting and we only sketch the proof: It can be checked that the conditional probability given by Proposition 2 is non-increasing in $k$ (the $\min$ operator is essential for this). From this the claim follows since for increasing $\bar{p}_n$ the binomial distribution puts more weight to terms with large $k$, thus not increasing the sum. $\square$

*Proof of Thm. 2.* The first three factors in the definition (1) of $\Delta(n, r, \bar{p}_n)$ are equal to a binomial probability $\text{Bin}(k \ ; \ n, \bar{p}_n)$, and the expectation of $k$ is thus $n\bar{p}_n$. By the Hoeffding bound, for all $\epsilon > 0$, the probability of $k < n(\bar{p}_n - \epsilon)$ is bounded by $\exp(-2n\epsilon^2)$. Applying this bound with $\epsilon = \bar{p}_n/3$ we get that the probability of $k < \frac{2}{3}\bar{p}_n$ is bounded by $\exp(-\frac{2}{9}n\bar{p}_n^2)$. Combined with (1) this gives the following upper bound on $\Delta(n, r, \bar{p}_n)$:

$$\exp\left(-\tfrac{2}{9}n\bar{p}_n^2\right)\max_{k < n\frac{2}{3}\bar{p}_n} f(n, r, k) + \max_{k \geq n\frac{2}{3}\bar{p}_n} f(n, r, k) \leq \exp\left(-\tfrac{2}{9}n\bar{p}_n^2\right) + \max_{k \geq n\frac{2}{3}\bar{p}_n} f(n, r, k) \tag{3}$$

where the maxima are taken over integer-valued $k$. In the last inequality we used the fact that for all $n, r, k$, it holds that $f(n, r, k) \leq 1$. Now note that for $k \geq r$, we can bound

$$f(n, r, k) \leq \binom{k}{r}\prod_{j=0}^{k-r-1}\frac{n-j}{n} \leq \binom{n}{r}\prod_{j=0}^{k}\frac{n-j}{n}\prod_{j=k-r}^{k}\frac{n}{n-j} \leq$$

$$\binom{n}{r}\prod_{j=1}^{k}\frac{n-j}{n}\left(\frac{n}{n-k}\right)^{r+1} \leq n^{2r}\frac{n}{n-k}\prod_{j=1}^{k}\frac{n-j}{n} \ . \tag{4}$$

If $k < r$, $f(n, r, k) = 1$ so that (4) holds in fact for all $k$ with $1 \leq k \leq n$. We bound the last factor $\prod_{j=1}^{k}\frac{n-j}{n}$ further as follows. The average of the $k$ factors of this product is less than or equal to $\frac{n-k/2}{n} = 1 - \frac{k}{2n}$. Since a product of $k$ factors is always less than or equal to the average of the factors to the power of $k$, we get the upper bound $\left(1 - \frac{k}{2n}\right)^k \leq \exp\left(-\frac{k \cdot k}{2n}\right) \leq \exp\left(-\frac{k^2}{2n}\right)$, where the first inequality follows from $1 - x \leq \exp(-x)$ for $x < 1$. Plugging this into (4) gives $f(n, r, k) \leq n^{2r}\frac{n}{n-k}\exp\left(-\frac{k^2}{2n}\right)$. Plugging this back into (3) gives $\Delta(n, r, \bar{p}_n) \leq \exp(-\frac{2}{9}n\bar{p}_n^2) + \max_{k \geq n\frac{2}{3}\bar{p}_n} 3n^{2r}\exp\left(-\frac{k^2}{2n}\right) \leq \exp(-\frac{2}{9}n\bar{p}_n^2) + 3n^{2r}\exp(-\frac{2}{9}n\bar{p}_n^2) \leq 4n^{2r}\exp(-\frac{2}{9}n\bar{p}_n^2)$.

Recall that $\mathcal{B}(\delta, D) := \arg\min_p \{p \ : \ \Delta(n, r, p) \leq \delta\}$. Replacing $\Delta(n, r, p)$ by the above upper bound, makes the set of $p$ satisfying the inequality smaller. Thus, the minimal member of the reduced set is greater than or equal to the minimal member of the set with $\Delta(n, r, p) \leq \delta$, giving the following bound on $\mathcal{B}(\delta, D)$:

$$\mathcal{B}(\delta, D) \leq \arg\min_p \left\{p \ : \ 4n^{2r}\exp\left(-\tfrac{2}{9}np^2\right) \leq \delta\right\} = 3\sqrt{\tfrac{1}{2n}\left(\log\tfrac{4}{\delta} + 2r\log n\right)} \ . \quad \square$$